# Learning in Compositional Hierarchies: Inducing the Structure of Objects from Data

**Joachim Utans**
Oregon Graduate Institute
Department of Computer Science and Engineering
P.O. Box 91000
Portland, OR 97291–1000
utans@cse.ogi.edu

## Abstract

I propose a learning algorithm for learning hierarchical models for object recognition. The model architecture is a compositional hierarchy that represents part-whole relationships: parts are described in the local context of substructures of the object. The focus of this report is learning hierarchical models from data, i.e. inducing the structure of model prototypes from observed exemplars of an object. At each node in the hierarchy, a probability distribution governing its parameters must be learned. The connections between nodes reflects the structure of the object. The formulation of substructures is encouraged such that their parts become conditionally independent. The resulting model can be interpreted as a *Bayesian Belief Network* and also is in many respects similar to the *stochastic visual grammar* described by Mjolsness.

## 1  INTRODUCTION

Model-based object recognition solves the problem of invariant recognition by relying on stored prototypes at unit scale positioned at the origin of an object-centered coordinate system. Elastic matching techniques are used to find a correspondence between features of the stored model and the data and can also compute the parameters of the transformation the observed instance has undergone relative to the stored model. An example is the TRAFFIC system (Zemel, Mozer and Hinton, 1990) or the Frameville system (Mjolsness, Gindi and

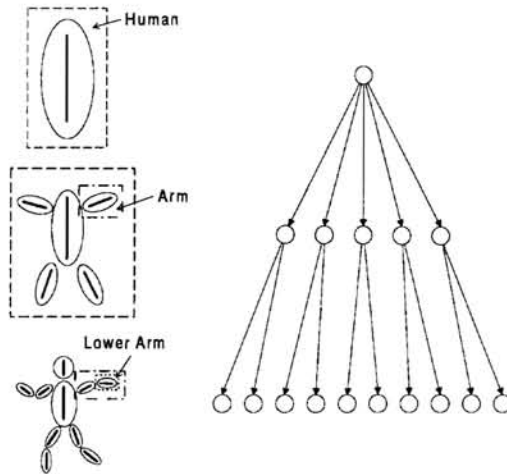

Figure 1: Example of a compositional hierarchy. The simple figure can be represented as hierarchical composition of parts. The hierarchy can be represented as a graph (a tree in this case). Nodes represent parts and edges represent the structural relationship. Nodes at the bottom represent individual parts of the object; nodes at higher levels denote more complex substructures. The single node at the top of the tree represents the entire object.

Anandan, 1989; Gindi, Mjolsness and Anandan, 1991; Utans, 1992). Frameville stores models as compositional hierarchies and by matching at each level in the hierarchy reduces the combinatorics of the match.

The attractive feature of feed-forward neural networks for object recognition is the relative ease with which their parameters can be learned from training data. Multilayer feed-forward networks are typically trained on input/output pairs (supervised learning) and thus are tuned to recognize instances of objects as seen during training. Difficulties arise if the observed object appears at a different position in the input image, is scaled or rotated, or has been subject to distortions. Some of these problems can be overcome by suitable preprocessing or judicious choice of features. Other possibilities are *weight sharing* (LeCun, Boser, Denker, Henderson, Howard, Hubbard and Jackel, 1989) or invariant distance measures (Simard, LeCun and Denker, 1993).

Few attempts have been reported in the neural network literature to learn the prototype models for model based recognition from data. For example, the Frameville system uses hand-designed models. However, models learned from data and reflecting the statistics of the data should be superior to the hand-designed models used previously. Segen (1988*a*; 1988*b*) reports an approach to learning structural descriptions where features are clustered to substructures using a *Minimum Description Length (MDL)* criterion to obtain a sparse representation. Saund (1993) has proposed a algorithm for constructing tree presentation with multiple "causes" where observed data is accounted for by multiple substructures at higher levels in the hierarchy. Ueda and Suzuki (1993) have developed an algorithm for learning models from shape contours using multiscale convex/concave structure matching to find a prototype shape typical for exemplars from a given class.

## 2   LEARNING COMPOSITIONAL HIERARCHIES

The algorithm described here merges parts by means of grouping variables to form substructures. The model architecture is a compositional hierarchy, i.e. a part-whole hierarchy (an example is shown in Figure 1). A prominent advocate of such models has been Marr (1982) and models of this type are used in the Frameville system (Mjolsness *et al.*, 1989; Gindi *et al.*, 1991; Utans, 1992). The nodes in the graph represent parts and substructures, the

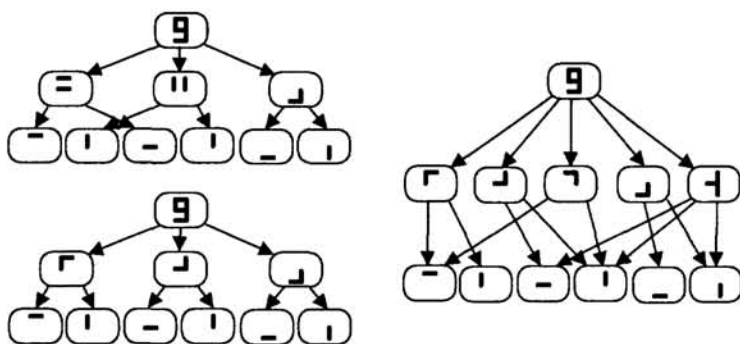

Figure 2: Examples of different compositional hierarchies for the same object (the digit 9 for a seven-segment LED display). One model emphasizes the parallel lines making up the square in the top part of the figure while for another model angles are chosen as intermediate substructures. The example on the right shows a hierarchy that "reuses" parts.

arcs describe the structure of the object. The arcs can be regarded as "part-of" or "ina" relationships (similar to the notion used in semantic networks). At each node a probability density for part parameters such as position, size and orientation is stored.

The model represents a typical prototype object at unit scale in an object-centered coordinate system. Parameters of parts are specified relative to parameters of the parent node in the hierarchy. Substructures thus provide a local context for their parts and decouple their parts from other parts and substructures in the model. The advantages of this representation are sparseness, invariance with respect to viewpoint transformations and the ability to model local deformations. In addition, the model explicitly represents the structure of an object and emphasizes the importance of structure for recognition (Cooper, 1989).

Learning requires estimating the parameters of the distributions at each node (the mean and variance in the case of Gaussians) and finding the structure of model. The emphasis in this report is on learning structure from exemplars. The parameterization of substructures may be different than for the parts at the lowest level and become more complex and require more parameters as the substructures themselves become more complex. The representation as compositional hierarchy can avoid overfitting since at higher levels in the hierarchy more exemplars are available for parameter estimation due to the grouping of parts (Omohundro, 1991).

## 2.1 Structure and Conditional Independence: Bayesian Networks

In what way should substructures be allocated? Figure 2 shows examples of different compositional hierarchies for the same object (the digit 9 for a seven-segment LED display). One model emphasizes the parallel lines making up the square in the top part of the figure while for another model angles are chosen as intermediate substructures. It is not clear which of these models to choose.

The important benefit of a hierarchical representation of structure is that parts belonging to different substructures become decoupled, i.e. they are assigned to a different local context. The problem of constructing structured descriptions of data that reflect this independence relationship has been studied previously in the field of Machine Learning (see (Pearl, 1988) for a comprehensive introduction). The resulting models are *Bayesian Belief Networks*. Central to the idea of Bayesian Networks is the assumption that objects can be regarded as being composed of components that only sparsely interact and the network captures the probabilistic dependency of these components. The network can be represented as an interaction graph augmented with conditional probabilities. The structure of the graph represents the dependence of variables, i.e. connects them with and arc. The strength of the

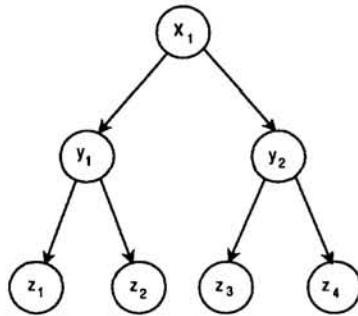

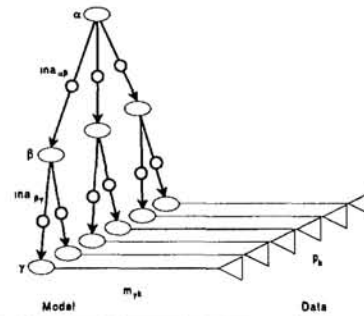

Figure 3: Bayesian Networks and conditional independence (see text).

Figure 4: The model architecture. Circles denote the grouping variables $ina$ (here a possible valid model after learning is shown).

dependence is expressed as forward conditional probability. The conditional independence is represented by the absence of an arc between two nodes and leads to the sparseness of the model.

The notion of conditional independence in the context studied here manifest itself as follows. By just observing two parts in the image, one must assume that they, i.e. their parameters such as position, are dependent and must be modeled using their joint distribution. However, if one knows that these two parts are grouped to form a substructure then knowing the parameters of the substructure, the parts become conditionally independent, namely conditioned on the parameters of the substructure. Thus, the internal nodes representing the substructures summarize the interaction of their child nodes. The correlation between the child nodes is summarized in the parent node and what remains is, for example, independent noise in observed instances of the child nodes.

The probability of observing an instance can be calculated from the model by starting at the root node and multiplying with the conditional probabilities of nodes traversed until the leaf nodes are reached. For example, given the graph in Figure 3, the joint distribution can be factored as

$$P(x_1, y_1, y_2, z_1, z_2, z_3, z_4) =$$
$$P(x_1)P(y_1|x_1)P(z_1|y_1)P(z_1|y_1)P(z_2|y_1)P(z_3|y_2)P(z_4|y_2) \qquad (1)$$

(note that the hidden nodes are treated just like the nodes corresponding to observable parts).

Note that the *stochastic visual grammar* described by Mjolsness (1991) is equivalent to this model. The model used there is a stochastic forward (generative) model where each level of the compositional hierarchy corresponds to a stochastic production rule that generates nodes in the next lower level. The distribution of parameters at the next lower level are conditioned on the parameters of the parent node. Thus, the model obtained from constructing a Bayesian network is equivalent to the stochastic grammar if the network is constrained to a directed acyclic graph (DAG).

If all the nodes of the network correspond to observable events, techniques exist for finding the structure of the Bayesian Network and estimate its parameters (Pearl, 1988) (see also (Cooper and Herskovits, 1992)). However, for the hierarchical models considered here, only the nodes at the lowest layer (the leaves of the tree) correspond to observable instances of parts of the object in the training data. The learning algorithm must induce hidden, unobservable substructures. That is, it is assumed that the observables are "caused" by internal nodes not directly accessible. These are represented as nodes in the network just

like the observables and their parameters must be estimated as well. See (Pearl, 1988) for an extensive discussion and examples of this idea.

Learning Bayesian networks is a hard problem when the network contains hidden nodes but a construction algorithm exists if it is known that the data is in fact tree-decomposable (Pearl, 1988). The methods is based on computing the correlations $\rho$ between child nodes and constraints on the correlation coefficients dictated by a particular structure. The entire tree can be constructed recursively using this method. Here, the case of Normal-distributed real-valued random variables is of interest:

$$p(x_1,\ldots,x_n) = \frac{1}{\sqrt{2\pi}^n} \frac{1}{\sqrt{\det \Sigma}} \exp\left(-\frac{1}{2}(\mathbf{x}-\mu)^T \Sigma^{-1}(\mathbf{x}-\mu)\right) \qquad (2)$$

where $\mathbf{x} = (x_1, x_2, \ldots, x_n)$ with mean $\mu = E\{\mathbf{x}\}$ and covariance matrix $\Sigma = E\{(\mathbf{x}-\mu)(\mathbf{x}-\mu)^T\}$ The method is based on a condition under which a set of random variables is *star-decomposable*. The question one ask is whether a set of $n$ random variables can be represented as the marginal distribution of $n+1$ variables $x_1,\ldots,x_n,w$ such that the $x_1,\ldots,x_n$ are conditionally independent given $w$, i.e.

$$p(x_1,\ldots,x_n,w) = \prod_i p(x_i|w)p(w) \qquad (3)$$

$$p(x_1,\ldots,x_n) = \int p(x_1,\ldots,x_n,w)dw \qquad (4)$$

In the graph representation of the Bayesian Network $w$ is the central node relating the $x_1,\ldots,x_n$, hence the name *star-decomposable*. In the general case of $n$ variables this is hard to verify but a result by Xu and Pearl (1987) is available for 3 variables: A necessary and sufficient condition for 3 random variables with a joint normal distribution to be star-decomposable is that the pairwise correlation coefficients satisfy the triangle inequality

$$\rho_{jk} \geq \rho_{ji}\rho_{ik} \qquad \text{with} \qquad \rho_{ij} = \frac{\sigma_{ij}}{\sqrt{\sigma_{ii}\sigma_{jj}}} \qquad (5)$$

for all $i,j,k \in [1,2,3]$ and $i \neq j \neq k$. Equality holds if node $w$ coincides with node $i$. For the lowest level of the hierarchy, nodes $j$ and $k$ represent parts and node $i = w$ represents the common substructure.

## 2.2   An Objective Function for Grouping Parts

The algorithm proposed here is based on "soft" grouping by means of grouping variables *ina* where both the grouping variables and the parameter estimates are updated concurrently. The learning algorithms described in (Pearl, 1988) incrementally construct a Bayesian network and decisions made at early stages cannot be reversed. It is hoped that the method proposed here is more robust with regard to inaccuracies of the estimates. However, if the true distribution is not a star-decomposable normal distribution it can only be approximated.

Let $ina_{ij}$ be a binary variable associated with the arc connecting node $i$ and node $j$; $ina_{ij} = 1$ if the arc is present in the network (**ina** is the adjacency matrix of the graph describing the structure of the model). The model architecture is restricted to a compositional hierarchy (a departure from the more general structure of a Bayesian Network, i.e. nodes are preassigned to levels of the hierarchy (see Figure 4)). Based on the condition in equation (5) a cost

function term for the grouping variables $ina$ is

$$E_\rho = \sum_{w,j,k \neq j} ina_{wj} ina_{wk} \left( \rho_{wj} \rho_{wk} - \rho_{jk} \right)^2 \tag{6}$$

The term penalizes the grouping of two part nodes to the same parent if the term in parentheses is large ($i$ and $k$ index part nodes, $w$ nodes at the next higher level in the hierarchy) The $ina_{wj}$ can be regarded as assignment variables the assign child nodes $j$ to parent nodes $w$. The parameters at each node and the assignment variables $ina$ are estimated using an EM algorithm (Dempster, Laird and Rubin, 1977; Utans, 1993; Yuille, Stolorz and Utans, 1994). For the details of the implementation of grouping with match networks see (Mjolsness et al., 1989; Mjolsness, 1991; Gindi et al., 1991; Utans, 1992; Utans, 1994).

At each node for each parameter a probability distribution is stored. Nodes at the lowest level of the hierarchy represent parts in the input data. For the Gaussian distributions used here for all nodes, the parameters are the mean $\mu$ and the variance $\sigma$ and can be estimated from data. Each part node can potentially be grouped to any substructure at the next higher level in the hierarchy. The parameters of the distributions at this level are estimated from data as well but using the current value of the grouping variables $ina_{ij}$ to weight the contribution from each part node. Because each child node $j$ can have only one parent node $i$, an additional constraint for a unique assignment is $\sum_w ina_{wj} = 1$.

## 3   AN EXAMPLE

Initial simulations of the proposed algorithm were performed using a hierarchial model for dot clusters. The training data was generated using the three-level model shown in Figure 5. Each node is parameterized by its position $(x, y)$. The node at the top level represents the entire dot cluster. At the intermediate level nodes represent subcluster centers. The leaf nodes at the lowest level represent individual dots that are output by the model and observed in the image. The top level node represents the position of the entire cluster. At each level $l + 1$ stored offsets $\mathbf{d}_{ij}^{l+1}$ are added to the parent coordinates $\mathbf{x}_i^l$ to obtain the coordinates of the child nodes. Then, independent, zero-mean Gaussian distributed noise $\epsilon$ is added: $\mathbf{x}_j^{l+1} = \mathbf{x}_i^l + \mathbf{d}_{ij}^{l+1} + \epsilon$ The training data consists of a vector of positions at the lowest level $\{\mathbf{x}_j\}$ with $\mathbf{x}_j = (x_j, y_j)$, $j = 1 \ldots 9$ for each exemplar.

The identity of the parts in the training data is assumed known. In addition, the data consists of parts from a single object. For the simulations, the model architecture is restricted to a three-level hierarchy. Since at the top level a single node represents the entire object, only the grouping variables from the lowest to the intermediate level are unknown (the nodes at the intermediate level are implicitly grouped to the single node at the top level). In the current implementation the parameters of a parent node are defined as the average over the parameters of its child nodes: $\hat{\mathbf{x}}_i^l = \frac{1}{N} \sum_j \widehat{ina_{ij}} \hat{\mathbf{x}}_j^{l+1}$

For this problem the algorithm has recovered the structure of the model that generated the training data. Thus in this case it is possible to use the correlation coefficients to learn the structure of an object from noisy training exemplars. However, the algorithm does not recover the same parameter values $\mathbf{x}$ used in the generative model at the intermediate layers. These cannot uniquely specified due to the ambiguity between the parameters $\mathbf{x}_i$ and offsets $\mathbf{d}_{ij}$ (a different choice for $\mathbf{x}_i$ leads to different values for $\mathbf{d}_{ij}$).

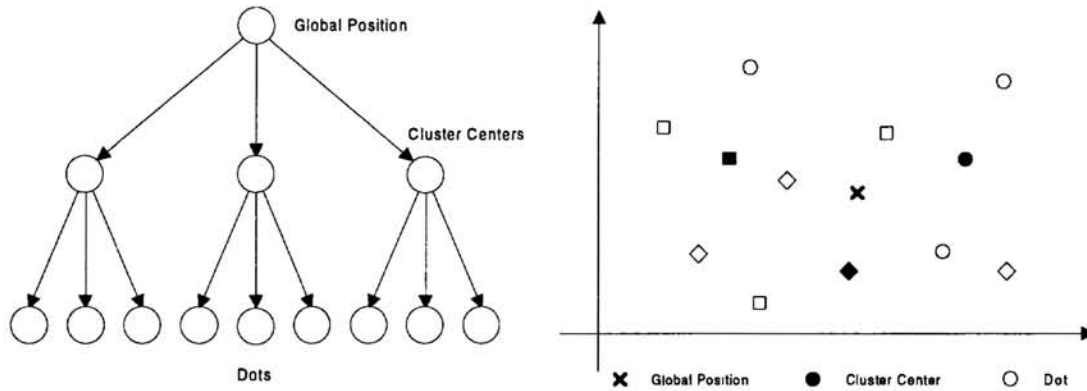

Figure 5: The model used to generated training data. The structure of the model is a three-level hierarchy. The model parameters are chosen such that the generated dot cluster spatially overlap. On the left, an example of an instance of a dot cluster generated from the model is shown (these constitute the training data).

## 4   EXTENSIONS

The results of the initial experiments are encouraging but more research needs to be done before the algorithm can be applied to real data. For the example used here, the training data was generated by a hierarchical model. Thus the distribution of the training exemplars could, in principle, be learned exactly using the proposed model architecture. I plan to study the effect of approximating the distribution of real-world data by applying the method to the problem of learning models for handwritten digit recognition.

The model should be extended to include provisions to deal with missing data. Instead of being binary variables, $ina_{ij}$ could be the conditional probability that part $j$ is present in a typical instance of the object given that the parent node $i$ itself is present (similar to the dot deletion rule described in (Mjolsness, 1991)). These probabilities must also be estimated from data. Under this interpretation the $ina_{ij}$ are similar to the mixture coefficients in the mixture of experts model (Jordan and Jacobs, 1993)

The robustness of the algorithm can be improved when the desired locality of the model is explicitly favored via an additional constraint.

$$E_{\text{local}} = \lambda \sum_{ijk} ina_{ij} ina_{ik} \left| \mathbf{x}_j - \mathbf{x}_k \right|^2$$

In this sense, the toy problem shown here is unnecessarily difficult. Preliminary experiments indicate that including this term reduces the sensitivity to spurious correlations between parts that are far apart.

As described the algorithm performs unsupervised grouping; learning the hierarchical model does not take in to account the recognition performance obtained when using the model. While the problem of learning and representing models in a hierarchical form is interesting in its own right, the final criteria for judging the model in the context of a recognition problem should be recognition performance. The assumption is that the model should pick up substructures that are specific to a particular class of objects and maximally discriminate between objects belonging to other classes. For example, after a initial model is obtained that roughly captures the structure of the training data, it can be refined on-line during the recognition stage.

## Acknowledgements

Initial work on this project was performed while the author was with the International Computer Science Institute, Berkeley, CA. At OGI supported was provided in part under grant ONR N00014-92-J-4062. Discussions with S. Knerr, E. Mjolsness and S. Omohundro were helpful in preparing this work.

## References

Cooper, G. F. and Herskovits, E. (1992), 'A bayesian method for induction of probabilistic networks from data', *Machine Learning* **9**, 309–347.

Cooper, P. R. (1989), Parallel Object Recognition from Structure (The Tinkertoy Project), PhD thesis, University of Rochester, Computer Science. also Technical Report No. 301.

Dempster, A. P., Laird, N. M. and Rubin, D. B. (1977), 'Maximum likelihood from incomplete data via the EM algorithm', *J. Royal Statist. Soc. B* **39**, 1–39.

Gindi, G., Mjolsness, E. and Anandan, P. (1991), Neural networks for model based recognition, *in* 'Neural Networks: Concepts, Applications and Implementations', Prentice–Hall, pp. 144–173.

Jordan, M. I. and Jacobs, R. A. (1993), Hierarchical mixtures of experts and the EM algorithm, Technical Report 9301, MIT Computational Cognitive Science.

LeCun, Y., Boser, B., Denker, J. S., Henderson, D., Howard, R. E., Hubbard, W. and Jackel, L. D. (1989), 'Backpropagation applied to handwritten zip code recognition', *Neural Computation* **1**, 541–551.

Marr, D. (1982), *Vision*, W. H. Freeman and Co., New York.

Mjolsness, E. (1991), Bayesian inference on visual grammars by neural nets that optimize, Technical Report YALEU–DCS–TR–854, Yale University, Dept. of Computer Science.

Mjolsness, E., Gindi, G. R. and Anandan, P. (1989), 'Optimization in model matching and perceptual organization', *Neural Computation* **1**(2).

Omohundro, S. M. (1991), Bumptrees for efficient function, constraint, and classification learning, *in* R. Lippmann, J. Moody and D. Touretzky, eds, 'Advances in Neural Information Processing 3', Morgan Kaufmann Publishers, San Mateo, CA.

Pearl, J. (1988), *Probabilistic Reasoning in Intelligent Systems: Networks of Plausible Inference*, Morgan Kaufmann Publishers, Inc., San Mateo, CA.

Saund, E. (1993), A multiple cause mixture model for unsupervised learning, Technical report, Xerox PARC, Palo Alto, CA. preprint, submitted to Neural Computation.

Segen, J. (1988a), Learning graph models of shape, *in* 'Proceedings of the 5th International Conference on Machine Learning'.

Segen, J. (1988b), 'Learning structural description of shape', *Machine Vision* pp. 257–269.

Simard, P., LeCun, Y. and Denker, J. (1993), Efficient pattern recognition using a new transformation distance, *in* S. J. Hanson, J. Cowan and L. Giles, eds, 'Advances in Neural Information Processing 5', Morgan Kaufmann Publishers, San Mateo, CA.

Ueda, N. and Suzuki, S. (1993), 'Learning visual models from shape contours using multiscale convex/concave structure matching', *IEEE Transactions on Pattern Analysis and Machine Intelligence* **15**(4), 337–352.

Utans, J. (1992), Neural Networks for Object Recognition within Compositional Hierarchies, PhD thesis, Department of Electrical Engineering, Yale University, New Haven, CT 06520.

Utans, J. (1993), Mixture models and the EM algorithm for object recognition within compositional hierarchies. part 1: Recognition, Technical Report TR-93-004, International Computer Science Institute, 1947 Center St., Berkeley, CA 94708.

Utans, J. (1994), 'Mixture models for learning and recognition in compositional hierarchies', *in preparation*.

Xu, L. and Pearl, J. (1987), Structuring causal tree models with continous variables, *in* 'Proceedings of the 3rd Workshop on Uncertainty in AI', pp. 170–179.

Yuille, A., Stolorz, P. and Utans, J. (1994), 'Statistical physics, mixtures of distributions and the EM algorithm', *to appear in* Neural Computation.

Zemel, R. S., Mozer, M. C. and Hinton, G. E. (1990), Traffic: Recognizing objects using hierarchical reference frame transformations, *in* D. S. Touretzky, ed., 'Advances in Neural Information Processing 2', Morgan Kaufman Pulishers, San Mateo, CA.
